# A Parallel Analog CCD/CMOS Signal Processor

**Charles F. Neugebauer**     **Amnon Yariv**
Department of Applied Physics
California Institute of Technology
Pasadena, CA 91125

## Abstract

A CCD based signal processing IC that computes a fully parallel single quadrant vector-matrix multiplication has been designed and fabricated with a $2\mu m$ CCD/CMOS process. The device incorporates an array of Charge Coupled Devices (CCD) which hold an analog matrix of charge encoding the matrix elements. Input vectors are digital with 1 - 8 bit accuracy.

## 1 INTRODUCTION

Vector-matrix multiplication (VMM) is often used in neural network theories to describe the aggregation of signals by neurons. An input vector encoding the activation levels of input neurons is multiplied by a matrix encoding the synaptic connection strengths to create an output vector. The analog VLSI architecture presented here has been devised to perform the vector-matrix multiplication using CCD technology. The architecture calculates a VMM in one clock cycle, an improvement over previous semiparallel devices (Agranat et al., 1988), (Chiang, 1990). This architecture is also useful for general signal processing applications where moderate resolution is required, such as image processing.

As most neural models have robust behavior in the presence of noise and inaccuracies, analog VLSI offers the potential for highly compact neural circuitry. Analog multiplication circuitry can be made much smaller than its digital equivalent, offering substantial savings in power and IC size at the expense of limited accuracy and programmability. Digital I/O, however, is desirable as it allows the use of standard memory and control circuits at the system level. The device presented here has digital input and analog output and elucidates all relevant performance characteristics including accuracy, speed, power dissipation and charge retention of the VMM. In practice, on-chip charge domain A/D converters are used for converting analog output signals to facilitate digital communication with off-chip devices.

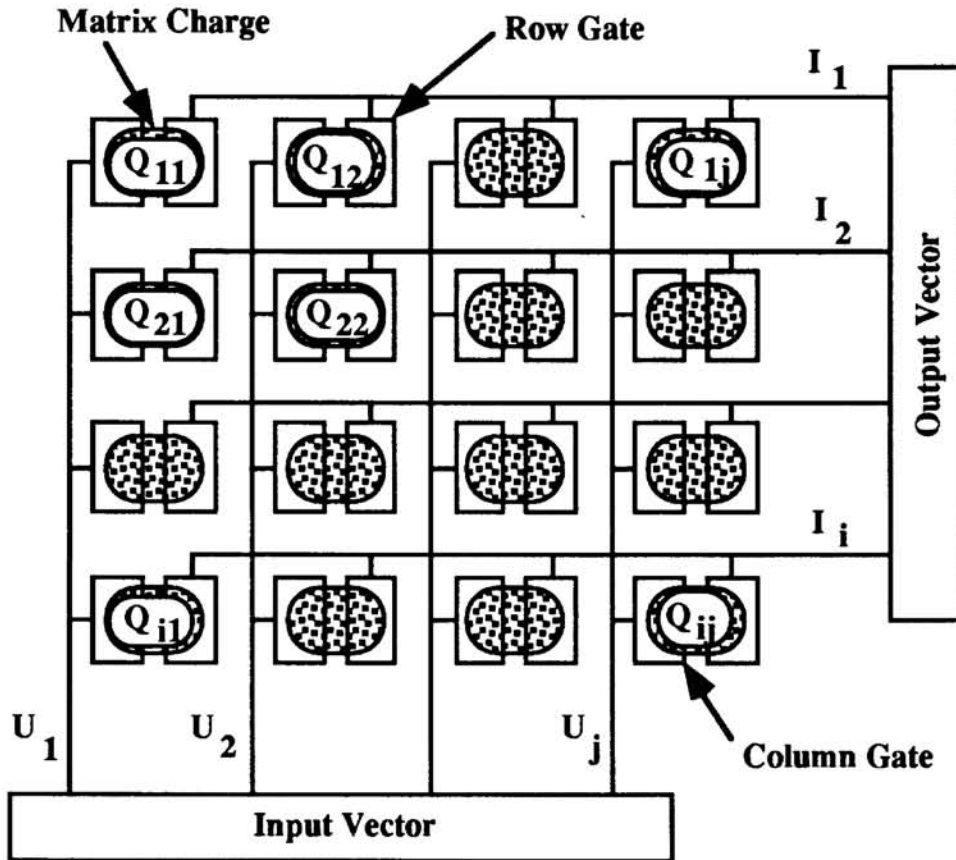

Figure 1: Simplified Schematic of CID Vector Matrix Multiplier

## 2 ARCHITECTURE DESCRIPTION

The vector-matrix multiplier consists of a matrix of CCD cells that resemble Charge Injection Device (CID) imager pixels in that one of the cell's gates is connected vertically from cell to cell forming a column electrode while another gate is connected horizontally forming a row electrode. The charge stored beneath the row and column gates encodes the matrix. A simplified schematic in Figure 1 shows the array organization.

### 2.1 BINARY VECTOR MATRIX MULTIPLICATION

In its most basic configuration, the VMM circuit computes the product of a binary input vector, $U_j$, and an analog matrix of charge. The computation done by each CID cell in the matrix is a multiply-accumulate in which the charge, $Q_{ij}$, is multiplied by a binary input vector element, $U_j$, encoded on the column line and this product is summed with other products in the same row to form the vector product, $I_i$, on the row lines. Multiplication by a binary number is equivalent to adding or not adding the charge at a

particular matrix element to its associated row line.

The matrix element operation is shown in Figure 2 which displays a cross-section of one of the rows with the associated potential wells at different times in the computation.

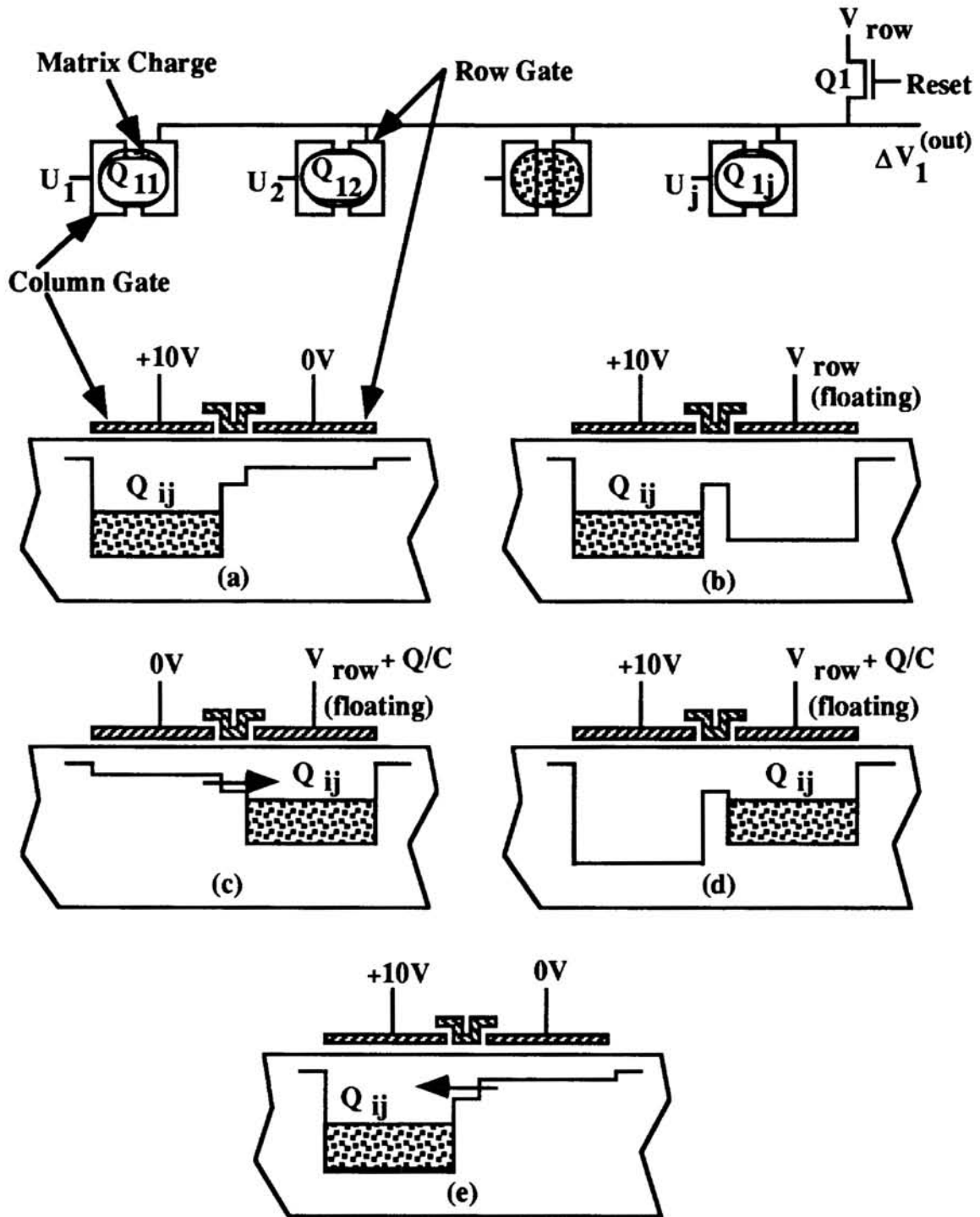

Figure 2: CID Cell Operation

In the initial state, prior to the VMM computation, the matrix of charges $Q_{ij}$ is moved

beneath the column electrodes by placing a positive voltage on all column lines, shown in Figure 2(a). A positive voltage creates a deep potential well for electrons. At this point, the row lines are reset to a reference voltage, $V_{row}$, by FETs Q1 and then disconnected from the voltage source, shown in Figure 2(b). The computation occurs when the column lines are pulsed to a negative voltage corresponding to the input vector $U_j$, shown in Figure 2(c). The binary $U_j$ is represented by a negative pulse on the $j^{th}$ column line if the element $U_j$ is a binary 1, otherwise the column line is kept at the positive voltage. This causes the charges in the columns that correspond to binary 1's in the input vector to be transferred to their respective row electrodes which thus experience a voltage change given by

$$\Delta V_i = \sum_{j=0}^{N-1} \frac{Q_{ij}U_j}{C_{row}}$$

where N is the number of elements in the input vector and $C_{row}$ is the total capacitance of the row electrode. Once the charge has been transferred, the column lines are reset to their original positive voltages[1], resulting in the potential diagram in Figure 2(d). The voltage changes on the row lines are then sampled and the matrix of charges are returned to the column electrodes in preparation for the next VMM by pulsing the row electrodes negative as in Figure 2(e). In this manner, a complete binary vector is multiplied by an analog matrix of charge in one CCD clock cycle.

# 3 DESIGN AND OPERATION

The implementation of this architecture contains facilities for electronic loading of the matrix. Originally proposed as an optically loaded device (Agranat et al., 1988), the electronically loaded version has proven more reliable and consistent.

## 3.1 LOADING THE CCD ARRAY WITH MATRIX ELEMENTS

The CCD matrix elements described above can be modified to operate as standard four phase CCD shift registers by simply adding another gate. The matrix cell is shown in Figure 3. The fabricated single quadrant cell size is 24μm by 24μm using a 2μm minimum feature size CCD/CMOS process. More aggressive design rules in the same process can reduce this to 20μm by 20μm. These cells, when abutted with each other in a row, form a horizontal shift register which is used to load the matrix. Electronic loading of the matrix is accomplished in a fashion similar to CCD imagers. A fast CCD shift register running vertically is added along one side of the matrix which is loaded with one column of matrix charges from a single external analog data source. Once the fast shift register is loaded, it is transferred into the array by clocking the matrix electrodes to act as an array of horizontal shift registers, shown in Figure 3(a). This process is repeated until the entire matrix has been filled with charge.

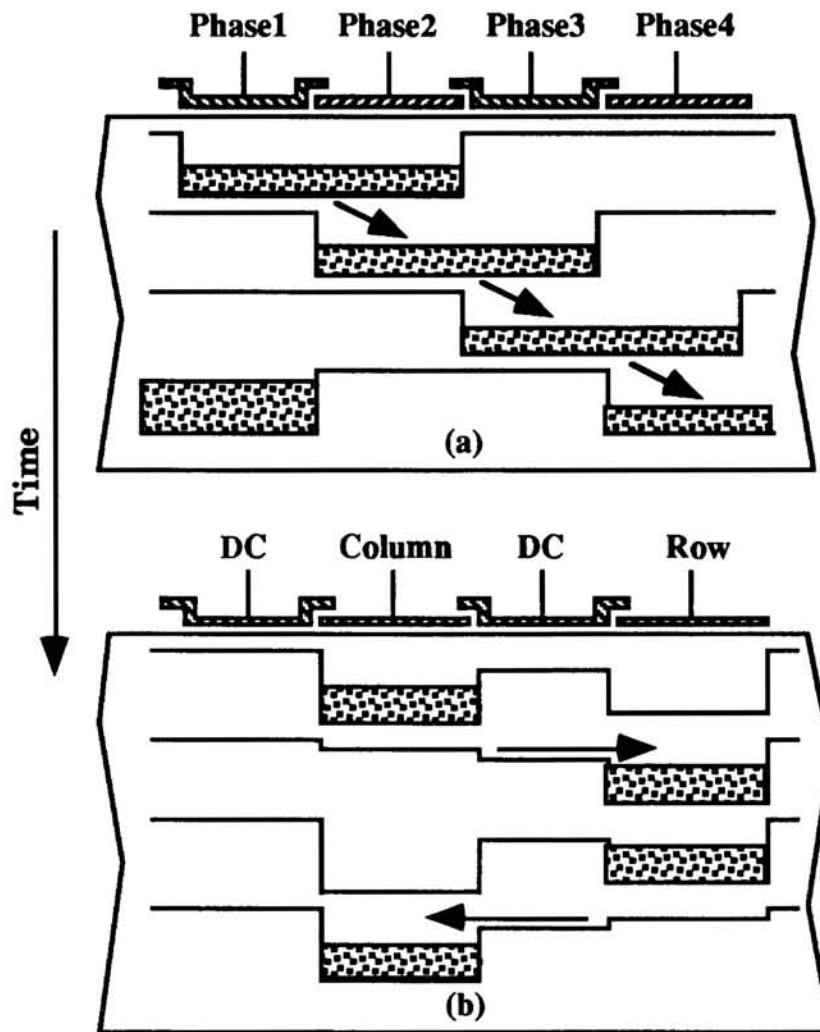

Figure 3: CID Cell Used to Load Matrix

When the matrix has been loaded, the charge can be used for computation with two of the four gates at each matrix cell kept at constant potentials, shown in Figure 3(b). The computation process moves the charge repeatedly between two electrodes. Incomplete charge transfer, a problem with our previous architecture (Agranat et al., 1990), does not degrade performance since any charge left behind under the column gates during computation is picked up on the next cycle, shown in Figure 2(e). Only dark current generation degrades the matrix charges during VMM, causing them to increase nonuniformly. In order to limit the effects of dark current generation on the matrix precision, the matrix charge must be refreshed periodically.

## 3.2 FLOATING GATE ROW AMPLIFIERS

In order to achieve better linearity when sensing charge, a floating gate amplifier is often used in CCD circuits. In the scheme described above, the induced voltage change of the row electrode significantly modifies its parasitic capacitance, resulting in a nonlinear voltage versus charge characteristic. To alleviate this problem, an operational amplifier with a capacitor in the feedback loop is added to each row line, shown in Figure 4. When

charge is moved underneath the row line in the course of a VMM operation, the row voltage is kept constant by the action of the op-amp with an output voltage given by

$$\Delta V_i = \sum_{j=0}^{N-1} \frac{Q_{ij}U_j}{C_f}$$

where $C_f$ is the feedback capacitance.

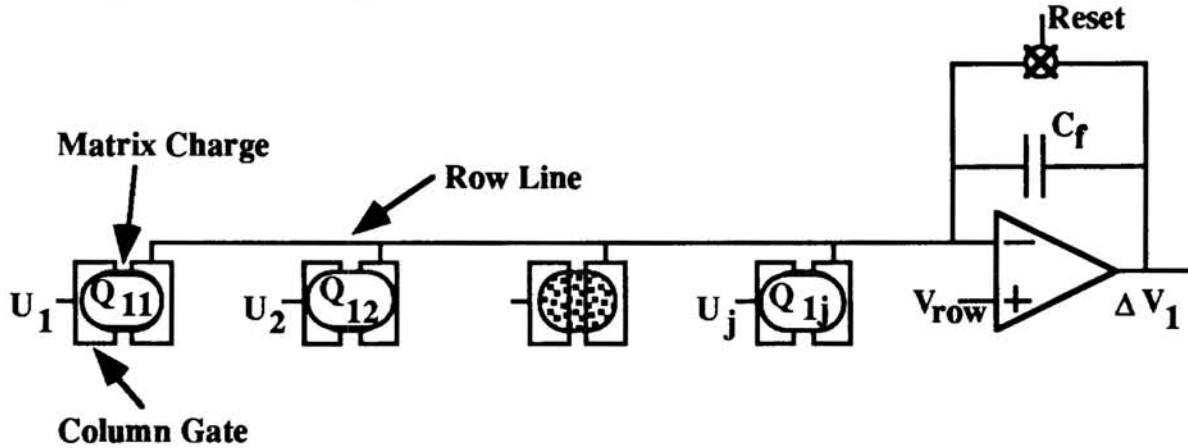

Figure 4: Linear Charge Sensing

The feedback capacitor is a poly-poly structure with vastly improved linearity compared to the row capacitance. This enhancement also has the effect of speeding the row line summation due to the well known benefits of current mode transmission. In addition, the possibility of digitally selecting a feedback capacitor value by switching power-of-two sized capacitors into the feedback loops creates a practical means of controlling the gain of the output amplifiers, with the potential for significantly extending the dynamic range of the device.

## 3.3 DIGITAL INPUT BUFFER AND DIVIDE-BY-TWO CIRCUITRY

Many applications such as image processing require multilevel input capability. This can easily be implemented by using the VMM circuitry in a bit-serial mode. The operation of the device is identical to the structure described above except that processing $n$-bit input precision requires $n$ cycles of the device. Digital shift registers are added to each input column line that sequentially present the column lines with successively more significant bits of the input vector, shown in Figure 5. Using the notation $U_j^{(n-1)}$, which represents the binary vector formed by taking the $n^{th}$ bits of all the input elements, the first VMM done by the circuit is given by

$$\Delta V_i^{(0)} = \sum_{j=0}^{N-1} \frac{Q_{ij}U_j^{(0)}}{C_f}$$

where $\Delta V_i^{(0)}$ is the output vector represented as voltage changes on the row lines. The row voltages are stored on large capacitors, C1, which are allowed to share charge with another set of equally sized capacitors, C2, effectively dividing the output vector by two.

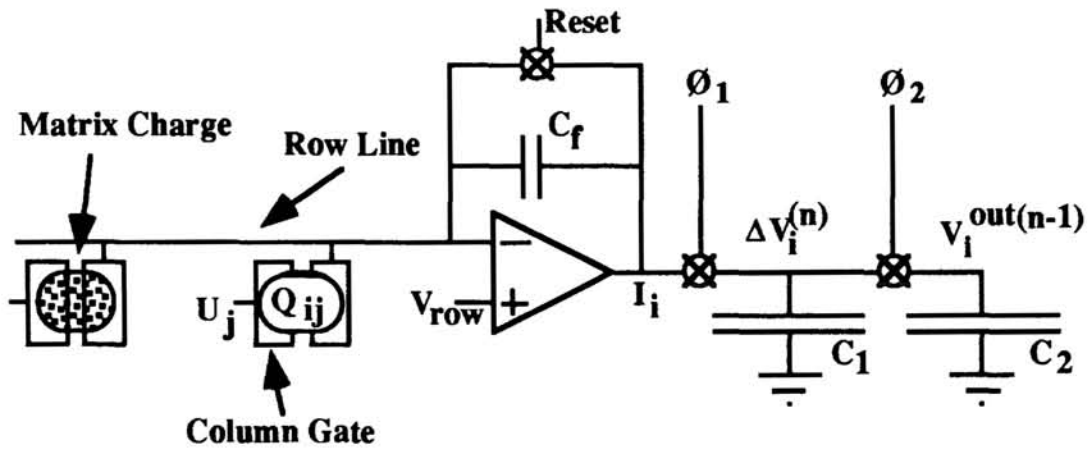

Figure 5: Switched Capacitor Divide-By-Two Circuit

The next most significant bit input vector, $U_j^{(1)}$, is then multiplied and creates another set of row voltage changes which are stored and shared to add another charge to the previously divided charge giving

$$V_i^{out\,(1)} = \sum_{j=0}^{N-1} \frac{Q_{ij}U_j^{(1)}}{C_f} + \frac{1}{2}\sum_{j=0}^{N-1} \frac{Q_{ij}U_j^{(0)}}{C_f}$$

where $V_i^{out(1)}$ is the voltage on C2 after two clock cycles. The process is repeated $n$ times, effectively weighting each successive bit's data by the proper power of two factor giving a total output voltage of

$$V_i^{out\,(n-1)} = \frac{1}{C_f}\left(\sum_{j=0}^{N-1} Q_{ij}\left(\sum_{k=1}^{n} 2^{k-n}U_j^{(n-1)}\right)\right) = \frac{1}{C_f}\sum_{j=0}^{N-1} Q_{ij}D_j$$

after $n$ clock cycles where $D_j$ now represents the multivalued digital input vector. In this manner, multivalued input of $n$-bit precision can be processed where $n$ is only limited by the analog accuracy of the components[2].

## 4  EXPERIMENTAL RESULTS

A number of VMM circuits have been fabricated implementing the architecture described above in a 2μm double-poly CCD/CMOS process. The largest circuit contains a 128x128 array of matrix elements. The matrix is loaded electronically through a single pin using the CCD shift register mode of the CID cell, shown in Figure 3. Matrix element mismatches due to threshold variations are avoided since all matrix elements are created by the same set of electrodes.

A list of relevant system characteristics is given in Table 1. The matrix of charge is

loaded in 4ms and needs to be refreshed every 20ms to retain acceptable weight accuracy at room temperature, giving a refresh overhead of 20%. A simple linear filter bank was loaded with a sinusoidal matrix and multiplied with a slowly chirped input signal to determine the linearity and noise limits.

### Table1: Experimental Results

| | |
|---|---|
| Charge Transfer Efficiency | 0.99995 |
| Cell Size | $24\mu m$ x $24\mu m$ |
| Bit Rate | 4 MHz |
| Refresh Time | 4ms |
| Noise Limits | 7 bits |
| Linearity | 5 bits |
| Power Consumption (excluding output drivers) | <100mW |
| Connections Per Second (binary input vectors) | $6.4 \times 10^{10}$ |

## 5  SUMMARY

A CCD based vector matrix multiplication scheme has been developed that offers high speed and low power in addition to provisions for digital I/O. Intended for neural network and image processing applications, the architecture is intended to integrate well into digital environments.

### Acknowledgements

This work was supported by a grant from the U.S. Army Center for Signals Warfare.

## Footnotes

[1]Returning the column lines to their original voltage levels has the effect of canceling the effect of stray capacitive coupling between the row and column lines, since the net column voltage change is zero.

[2] If 4-bit input is required the device is simply clocked four times. Since the power of two scaling is divisive, the most significant bit is always given the same weighting regardless of the input word length.

### References

A. Agranat, C. F. Neugebauer and A. Yariv. (1988) Parallel Optoelectronic Realization of Neural Network Models Using CID Technology. *Applied Optics 27* :4354-4355.

A. Agranat, C. F. Neugebauer, R.D. Nelson and A. Yariv. (1990) The CCD Neural Processor: A Neural Integrated Circuit with 65,536 Programmable Analog Synapses. *IEEE Trans. on Circuits and Systems 37* :1073-1075.

A. M. Chiang. (1990) A CCD Programmable Signal Processor. *IEEE Journal of Solid State Circuits 25* :1510-1517.